# Large Scale Hidden Semi-Markov SVMs

**Gunnar Rätsch**[*]
Friedrich Miescher Laboratoy, Max Planck Society
Spemannstr. 39, 72070 Tübingen, Germany
`Gunnar.Raetsch@tuebingen.mpg.de`

**Sören Sonnenburg**
Fraunhofer FIRST.IDA
Kekuléstr. 7, 12489 Berlin, Germany
`sonne@first.fhg.de`

## Abstract

We describe Hidden Semi-Markov Support Vector Machines (SHM SVMs), an extension of HM SVMs to semi-Markov chains. This allows us to predict segmentations of sequences based on segment-based features measuring properties such as the length of the segment. We propose a novel technique to partition the problem into sub-problems. The independently obtained partial solutions can then be recombined in an efficient way, which allows us to solve label sequence learning problems with several thousands of labeled sequences. We have tested our algorithm for predicting gene structures, an important problem in computational biology. Results on a well-known model organism illustrate the great potential of SHM SVMs in computational biology.

## 1  Introduction

Hidden Markov SVMs are a recently-proposed method for predicting a label sequence given the input sequence [3, 17, 18, 1, 2]. They combine the benefits of the power and flexibility of kernel methods with the idea of Hidden Markov Models (HMM) [11] to predict label sequences. In this work we introduce a generalization of Hidden Markov SVMs, called Hidden Semi-Markov SVMs (HSM SVMs). In HM SVMs and HMMs there is a state transition for every input symbol. In semi-Markov processes it is allowed to persist in a state for a number of time steps before transitioning into a new state. During this segment of time the system's behavior is allowed to be non-Markovian. This adds flexibility for instance to model segment lengths or to use non-linear content sensors that may depend on the start *and* end of the segment.

One of the largest problems with HM SVMs and also SHM SVMs is their high computational complexity. Solving the resulting optimization problems may become computationally infeasible already for a few hundred examples. In the second part of the paper we consider the case of using *content sensors* (for whole segments) and *signal detectors* (at segment boundaries) in SHM SVMs. We motivate a simple, but very effective strategy of partitioning the problem into independent sub-problems and discuss how one can reunion the different parts. We propose to solve a relatively small optimization problem that can be solved rather efficiently. This strategy allows us to tackle significantly larger label sequence problems (with several thousands of sequences).

To illustrate the strength of our approach we have applied our algorithm to an important problem in computational biology: the prediction of the segmentation of a pre-mRNA sequence into exons and introns. On problems derived from sequences of the model organism *Caenorhabditis elegans* we can show that the SHM SVM approach consistently outperforms HMM based approaches by a large margin (see also [13]).

The paper is organized as follows: In Section 2 we introduce the necessary notation, HM SVMs and the extension to semi-Markov models. In Section 3 we propose and discuss a technique that allows us to train SHM SVMs on significantly more training examples. Finally, in Section 4 we outline the gene structure prediction problem, discuss additional techniques to apply SHM SVMs to this problem and show surprisingly large improvements compared to state-of-the-art methods.

---

[*]Corresponding author, `http://www.fml.mpg.de/raetsch`

## 2 Hidden Markov SVMs

In label sequence learning one learns a function that assigns to a sequence of objects $\boldsymbol{x} = \chi_1\chi_2\ldots\chi_l$ a sequence of labels $\boldsymbol{y} = \upsilon_1\upsilon_2\ldots\upsilon_l$ ($\chi_i \in X$, $\upsilon_i \in \Upsilon$, $i = 1,\ldots,l$). While objects can be of rather arbitrary kind (e.g. vectors, letters, etc), the set of labels $\Upsilon$ has to be finite.[1] A common approach is to determine a discriminant function $F : \mathcal{X} \times \mathcal{Y} \to \mathbb{R}$ that assigns a score to every input $\boldsymbol{x} \in \mathcal{X} := X^*$ and every label sequence $\boldsymbol{y} \in \mathcal{Y} := \Upsilon^*$, where $X^*$ denotes the Kleene closure of $X$. In order to obtain a prediction $f(\boldsymbol{x}) \in \mathcal{Y}$, the function is maximized with respect to the second argument:

$$f(\boldsymbol{x}) = \operatorname*{argmax}_{\boldsymbol{y}\in\mathcal{Y}} F(\boldsymbol{x},\boldsymbol{y}). \tag{1}$$

### 2.1 Representation & Optimization Problem

In Hidden Markov SVMs (HM SVMs) [3], the function $F(\boldsymbol{x},\boldsymbol{y}) := \langle \boldsymbol{w}, \Phi(\boldsymbol{x},\boldsymbol{y})\rangle$ is linearly parametrized by a weight vector $\boldsymbol{w}$, where $\Phi(\boldsymbol{x},\boldsymbol{y})$ is some mapping into a feature space $\mathcal{F}$. Given a set of training examples $(\boldsymbol{x}_n,\boldsymbol{y}_n), n = 1,\ldots,N$, the parameters are tuned such that the true labeling $\boldsymbol{y}_n$ scores higher than all other labelings $\boldsymbol{y} \in \mathcal{Y}_n := \mathcal{Y} \setminus \boldsymbol{y}_n$ with a large margin, i.e. $F(\boldsymbol{x}_n,\boldsymbol{y}_n) \gg \operatorname{argmax}_{\boldsymbol{y}\in\mathcal{Y}_n} F(\boldsymbol{x}_n,\boldsymbol{y})$. This goal can be achieved by solving the following optimization problem (appeared equivalently in [3]):

$$\min_{\boldsymbol{\xi}\in\mathbb{R}^N, \boldsymbol{w}\in\mathcal{F}} \quad C\sum_{n=1}^{N} \xi_i + \boldsymbol{P}(\boldsymbol{w}) \tag{2}$$

$$\text{s.t.} \quad \langle\boldsymbol{w},\Phi(\boldsymbol{x},\boldsymbol{y}_n)\rangle - \langle\boldsymbol{w},\Phi(\boldsymbol{x},\boldsymbol{y})\rangle \geq 1 - \xi_n \quad \text{for all } n = 1,\ldots,N \text{ and } \boldsymbol{y} \in \mathcal{Y}_n,$$

where $\boldsymbol{P}$ is a suitable regularizer (e.g. $\boldsymbol{P}(\boldsymbol{w}) = \|\boldsymbol{w}\|^2$) and the $\xi$'s are slack variables to implement a soft margin. Note that the linear constraints in (2) are equivalent to the following set of nonlinear constraints: $F(\boldsymbol{x}_n,\boldsymbol{y}_n) - \max_{\boldsymbol{y}\in\mathcal{Y}_n} F(\boldsymbol{x}_n,\boldsymbol{y}) \geq 1 - \xi_n$ for $n = 1,\ldots,N$ [3].

If $P(\boldsymbol{w}) = \|\boldsymbol{w}\|^2$, it can be shown that the solution $\boldsymbol{w}^*$ of (2) can be written as

$$\boldsymbol{w}^* = \sum_{n=1}^{N}\sum_{\boldsymbol{y}\in\mathcal{Y}} \alpha_n(\boldsymbol{y})\Phi(\boldsymbol{x}_n,\boldsymbol{y}),$$

where $\alpha_n(\boldsymbol{y})$ is the Lagrange multiplier of the constraint involving example $n$ and labeling $\boldsymbol{y}$ (see [3] for details). Defining the kernel as $k((\boldsymbol{x},\boldsymbol{y}),(\boldsymbol{x}',\boldsymbol{y}')) := \langle\Phi(\boldsymbol{x},\boldsymbol{y}),\Phi(\boldsymbol{x}',\boldsymbol{y}')\rangle$, we can rewrite $F(\boldsymbol{x},\boldsymbol{y})$ as

$$F(\boldsymbol{x}',\boldsymbol{y}') = \sum_{n=1}^{N}\sum_{\boldsymbol{y}\in\mathcal{Y}} \alpha_n(\boldsymbol{y})k((\boldsymbol{x}_n,\boldsymbol{y}),(\boldsymbol{x}',\boldsymbol{y}')).$$

### 2.2 Outline of an Optimization Algorithm

The number of constraints in (2) can be very large, which may constitute challenges for efficiently solving problem (2). Fortunately, only a few of the constraints usually are active and working set methods can be applied in order to solve the problem for larger number of examples. The idea is to start with small sets of negative (i.e. false) labelings $\overline{\mathcal{Y}}_n$ for every example. One solves (2) for the smaller problem and then identifies labelings $\boldsymbol{y} \in \mathcal{Y}_n$ that maximally violate constraints, i.e.

$$\boldsymbol{y} = \operatorname*{argmax}_{\boldsymbol{y}\in\mathcal{Y}_n} F(\boldsymbol{x}_n,\boldsymbol{y}), \tag{3}$$

where $\boldsymbol{w}$ is the intermediate solution of the restricted problem. The new constraint generated by the negative labeling is then added to the optimization problem. The method described above is also known as column generation method or cutting-plane algorithm and can be shown to converge to the optimal solution $\boldsymbol{w}^*$ [18]. However, since the computation of $F$ involves many kernel computations and also the number of non-zero $\alpha$'s is often large, solving the problem with more than a few hundred labeled sequences often seems computationally too expensive.

### 2.3 Viterbi-like Decoding

Determining the optimal labeling in (1) efficiently is crucial during optimization and prediction. If $F(\boldsymbol{x},\cdot)$ satisfies certain conditions, one can use a Viterbi-like algorithm [20] for efficient decoding

of the optimal labeling. This is particularly the case when $\Phi$ can be written as a sum over the length of the sequence and decomposed as

$$\Phi(\boldsymbol{x}, \boldsymbol{y}) = \left( \sum_{i=1}^{l(\boldsymbol{x})} \Phi_{\sigma,\tau}(v_i, v_{i+1}, \boldsymbol{x}, i) \right)_{\sigma,\tau \in \Upsilon}$$

where $l(\boldsymbol{x})$ is the length of the sequence $\boldsymbol{x}$.[2] By $(\phi_\gamma)_{\gamma \in \Gamma}$ we denote the concatenation of feature vectors, i.e. $(\phi_{\gamma_1}^\top, \phi_{\gamma_2}^\top, \dots)^\top$. It is essential that $\Phi$ is composed of mapping functions that depend only on labels at position $i$ and $i+1$, $\boldsymbol{x}$ as well as $i$. We can rewrite $F$ using $\boldsymbol{w} = (\boldsymbol{w}_{\sigma,\tau})_{\sigma,\tau \in \Upsilon}$:

$$F(\boldsymbol{x}, \boldsymbol{y}) = \sum_{\sigma,\tau \in \Upsilon} \left\langle \boldsymbol{w}_{\sigma,\tau}, \sum_{i=1}^{l(\boldsymbol{x})} \Phi_{\sigma,\tau}(v_i, v_{i+1}, \boldsymbol{x}, i) \right\rangle = \sum_{i=1}^{l(\boldsymbol{x})} \underbrace{\sum_{\sigma,\tau \in \Upsilon} \langle \boldsymbol{w}_{\sigma,\tau}, \Phi_{\sigma,\tau}(v_i, v_{i+1}, \boldsymbol{x}, i) \rangle}_{=:g(v_i, v_{i+1}, \boldsymbol{x}, i)}. \quad (4)$$

Thus we have positionally decomposed the function $F$. The score at position $i+1$ only depends on $\boldsymbol{x}$, $i$ and labels at positions $i$ and $i+1$ (Markov property).

Using this decomposition we can define

$$V(i, v) := \begin{cases} \max_{v' \in \Upsilon}(V(i-1, v') + g(v', v, \boldsymbol{x}, i-1)) & i > 1 \\ 0 & \text{otherwise} \end{cases}$$

as the maximal score for all labelings with label $v$ at position $i$. Via dynamic programming one can compute $\max_{v \in \Upsilon} V(l(\boldsymbol{x}), v)$, which can be proven to solve (1) for the considered case. Moreover, using backtracking one can recover the optimal label sequence.[3]

The above decoding algorithm requires to evaluate $g$ at most $|\Upsilon|^2 l(\boldsymbol{x})$ times. Since computing $g$ involves computing potentially large sums of kernel functions, the decoding step can be computationally quite demanding–depending on the kernels and the number of examples.

## 2.4 Extension to Hidden Semi-Markov SVMs

Semi-Markov models extend hidden Markov models by allowing each state to persist for a non-unit number $\delta_i$ of symbols. Only after that the system will transition to a new state, which only depends on $\boldsymbol{x}$ and the current state. During the interval $(i, i + \delta_i)$ the behavior of the system may be non-Markovian [14]. Semi-Markov models are fairly common in certain applications of statistics [6, 7] and are also used in reinforcement learning [16]. Moreover, [15, 9] previously proposed an extension of HMMs, called Generalized HMMs (GHMMs) that is very similar to the ideas above. Also, [14] proposed a semi-Markov extension to Conditional Random Fields.

In this work we extend Hidden Markov-SVMs to Hidden Semi-Markov SVMs by considering sequences of segments instead of simple label sequences. We need to extend the definition of the labeling with $s$ segments: $\boldsymbol{y} = (v_1, \pi_1), (v_2, \pi_2), \dots, (v_s, \pi_s)$, where $\pi_j$ is the start position of the segment and $v_j$ its label.[4] We assume $\pi_1 = 1$ and let $\pi_j = \pi_{j-1} + \delta_j$. To simplify the notation we define $\pi_{s+1} := l(\boldsymbol{x}) + 1$, $s := s(\boldsymbol{y})$ to be the number of segments in $\boldsymbol{y}$ and $v_{s+1} := \emptyset$. We can now generalize the mapping $\Phi$ to:

$$\Phi(\boldsymbol{x}, \boldsymbol{y}) = \left( \sum_{j=1}^{s(\boldsymbol{y})} \Phi_{\sigma,\tau}(v_j, v_{j+1}, \boldsymbol{x}, \pi_j, \pi_{j+1}) \right)_{\sigma,\tau \in \Upsilon}.$$

$$V(i, v, k) := \begin{cases} \max_{v' \in \Upsilon, k'=1,\dots,K}^{(k)} (V(i-1, v', k') + g(v', v, \boldsymbol{x}, i-1)) & i > 1 \\ 0 & \text{otherwise} \end{cases}$$

where $\max^{(k)}$ is the function computing the $k^{\text{th}}$ largest number and is $-\infty$ if there are fewer numbers. $V(i, v, k)$ now is the $k$-best score of labelings with label $v$ at position $i$.

[4] For simplicity, we associate the label of a segment with the signal at the boundary to the next segment. A generalization is straightforward.

With this definition we can extract features from segments: As $\pi_j$ and $\pi_{j+1}$ are given one can for instance compute the length of the segment or other features that depend on the start *and* the end of the segment. Decomposing $F$ results in:

$$F(\boldsymbol{x}, \boldsymbol{y}) \quad = \quad \sum_{j=1}^{s(\boldsymbol{y})} \underbrace{\sum_{\sigma, \tau \in \Upsilon} \langle \boldsymbol{w}_{\sigma, \tau}, \Phi_{\sigma, \tau}(v_j, v_{j+1}, \boldsymbol{x}, \pi_j, \pi_{j+1}) \rangle}_{=:g(v_j, v_{j+1}, \boldsymbol{x}, \pi_j, \pi_{j+1})}. \tag{5}$$

Analogously we can extend the formula for the Viterbi-like decoding algorithm [14]:

$$V(i, v) := \begin{cases} \max\limits_{v' \in \Upsilon, d=1, \ldots, \min(i-1, S)} (V(i-d, v') + g(v', v, \boldsymbol{x}, i-d, i)) & i > 1 \\ 0 & \text{otherwise} \end{cases} \tag{6}$$

where $S$ is the maximal segment length and $\max_{v \in \Upsilon} V(l(\boldsymbol{x}), v)$ is the score of the best segment labeling. The function $g$ needs to be evaluated at most $|\Upsilon|^2 l(\boldsymbol{x}) S$ times. The optimal label sequence can be obtained as before by backtracking. Also the above method can be easily extended to produce the $K$ best labelings (cf. Footnote 3).

## 3 An Algorithm for Large Scale Learning

### 3.1 Preliminaries

In this section we consider a specific case that is relevant for the application that we have in mind. The idea is that the feature map should contain information about segments such as the length or the *content* as well as segment boundaries, which may exhibit certain detectable *signals*. For simplicity we assume that it is sufficient to consider the string $\chi_{\pi_j..\pi_{j+1}} := \chi_{\pi_j} \chi_{\pi_j+1} \cdots \chi_{\pi_{j+1}-2} \chi_{\pi_{j+1}-1}$ for extracting content information about segment $j$. Also, for considering signals we assume it to be sufficient to consider a window $\pm \omega$ around the end of the segment, i.e. we only consider $\chi_{\pi_{j+1} \pm \omega} := \chi_{\pi_{j+1}-\omega} \cdots \chi_{\pi_{j+1}+\omega}$. To keep the notation simple we do not consider signals at the start of the segment. Moreover, we assume for simplicity that $\boldsymbol{x}_{\pi \pm \omega}$ is appropriately defined for every $\pi = 1, \ldots, l(\boldsymbol{x})$. We may therefore define the following feature map:

$$\Phi(\boldsymbol{x}, \boldsymbol{y}) = \begin{pmatrix} \left( \sum\limits_{j=1}^{s(\boldsymbol{y})} [\![ v_j = \sigma ]\!] [\![ v_{j+1} = \tau ]\!] \Phi_c(\chi_{\pi_j..\pi_{j+1}}) \right)_{\sigma, \tau \in \Upsilon} & \quad \%content \\ \left( \sum\limits_{j=1}^{s(\boldsymbol{y})} [\![ v_{j+1} = \tau ]\!] \Phi_s(\chi_{\pi_{j+1} \pm \omega}) \right)_{\tau \in \Upsilon} & \quad \%signal \end{pmatrix}$$

where $[\![ true ]\!] = 1$ and 0 otherwise. Then the kernel between two examples using this feature map can be written as:

$$k((\boldsymbol{x}, \boldsymbol{y}), (\boldsymbol{x}', \boldsymbol{y}')) = \sum_{\substack{\sigma, \tau \in \Upsilon}} \sum_{\substack{j:(v_j, v_j)=(\sigma, \tau) \\ j':(v'_{j'}, v'_{j'})=(\sigma, \tau)}} k_c(\chi_{\pi_j..\pi_{j+1}}, \chi'_{\pi'_{j'}..\pi'_{j'+1}}) + \sum_{\tau \in \Upsilon} \sum_{\substack{j:v_{j+1}=\tau \\ j':v'_{j'+1}=\tau}} k_s(\chi_{\pi_{j+1} \pm \omega}, \chi'_{\pi'_{j'+1} \pm \omega})$$

where $k_c(\cdot, \cdot) := \langle \Phi_1(\cdot), \Phi_1(\cdot) \rangle$ and $k_s(\cdot, \cdot) := \langle \Phi_s(\cdot), \Phi_s(\cdot) \rangle$. The above formulation has the benefit of keeping the signals and content kernels separated for each label, which we can exploit for rewriting $F(\boldsymbol{x}, \boldsymbol{y})$

$$F(\boldsymbol{x}, \boldsymbol{y}) = \sum_{\sigma, \tau \in \Upsilon} \sum_{j:(v_j, v_{j+1})=(\sigma, \tau)} F_{\sigma, \tau}(\chi_{\pi_j..\pi_{j+1}}) + \sum_{\tau \in \Upsilon} \sum_{j:v_{j+1}=\tau} F_\tau(\chi_{\pi_{j+1} \pm \omega}),$$

where

$$F_{\sigma, \tau}(\boldsymbol{\chi}) := \sum_{n=1}^{N} \sum_{\boldsymbol{y}' \in \mathcal{Y}} \alpha_n(\boldsymbol{y}') \sum_{j':(v'_{j'}, v'_{j'+1})=(\sigma, \tau)} k_c(\boldsymbol{\chi}, \chi^n_{\pi'_{j'}..\pi'_{j'+1}})$$

and

$$F_\tau(\boldsymbol{\chi}) = \sum_{n=1}^{N} \sum_{\boldsymbol{y}' \in \mathcal{Y}} \alpha_n(\boldsymbol{y}') \sum_{j':v_{j'+1}=\tau} k_s(\boldsymbol{\chi}, \chi^n_{\pi'_{j'+1} \pm \omega}).$$

Hence, we have partitioned $F(\boldsymbol{x}, \boldsymbol{y})$ into $|\Upsilon|^2 + |\Upsilon|$ functions characterizing the content and the signals.

## 3.2 Two-Stage Learning

By enumerating all non-zero $\alpha$'s and valid settings of $j'$ in $F_\tau$ and $F_{\sigma,\tau}$, we can define sets of sequences $\{\boldsymbol{\chi}_m^{\tau,\sigma}\}_{m=1,\ldots,M_{\sigma,\tau}}$ and $\{\boldsymbol{\chi}_m^\tau\}_{m=1,\ldots,M_\tau}$ where every element is of the form $\chi_{\pi_j..\pi_{j+1}}^n$ and $\chi_{\pi_{j+1}\pm\omega}^n$, respectively. Hence, $F_\tau$ and $F_{\sigma,\tau}$ can be rewritten as a (single-sum) linear combination of kernels: $F_{\sigma,\tau}(\boldsymbol{\chi}) := \sum_{m=1}^{M_{\sigma,\tau}} \alpha_m^{\sigma,\tau} k_c(\boldsymbol{\chi}, \boldsymbol{\chi}_m^{\tau,\sigma})$ and $F_\tau(\boldsymbol{\chi}) := \sum_{m=1}^{M_\tau} \alpha_m^\tau k_s(\boldsymbol{\chi}, \boldsymbol{\chi}_m^\tau)$ for appropriately chosen $\alpha$'s. For sequences $\boldsymbol{\chi}_m^\tau$ that do not correspond to true segment boundaries, the coefficient $\alpha_m^\tau$ is either negative or zero (since wrong segment boundaries can only appear in wrong labelings $\boldsymbol{y} \neq \boldsymbol{y}_n$ and $\alpha_n(\boldsymbol{y}) \leq 0$). True segment boundaries in correct label sequences have non-negative $\alpha_m^\tau$'s. Analogously with segments $\boldsymbol{\chi}_m^{\tau,\sigma}$. Hence, we may interpret these functions as SVM classification functions recognizing segments and boundaries of all kinds.

Hidden Semi-Markov SVMs simultaneously optimize all these functions and also determine the relative importance of the different signals and sensors. In this work we propose to separate the learning of the content sensors and signal detectors from learning how they have to act together in order to produce the correct labeling. The idea is to train SVM-based classifiers $\bar{F}_{\sigma,\tau}$ and $\bar{F}_\tau$ using the kernels $k_c$ and $k_s$ on examples with known labeling. For every segment type and segment boundary we generate a set of positive examples from observed segments and boundaries. As negative examples we use all boundaries and segments that were not observed in a true labeling. This leads to a set of sequences that may potentially also appear in the expansions of $F_{\sigma,\tau}$ and $F_\tau$. However, the expansion coefficients $\bar{\alpha}_m^{\sigma;\tau}$ and $\bar{\alpha}_m^\tau$ are expected to be different as the functions are estimated independently.

The advantage of this approach is that solving two-class problems–for which we can reuse existing large scale learning methods–is much easier than solving the full HSM SVM problem. However, while the functions $\bar{F}_{\sigma,\tau}$ and $\bar{F}_\tau$ might recognize the same contents and signals as $F_{\sigma,\tau}$ and $F_\tau$, the functions are obtained independently from each other and might not be scaled correctly to jointly produce the correct labeling. We therefore propose to learn transformations $t_{\sigma,\tau}$ and $t_\tau$ such that $F_{\sigma,\tau}(\boldsymbol{\chi}) \approx t_{\sigma,\tau}(\bar{F}_{\sigma,\tau}(\boldsymbol{\chi}))$ and $F_\tau(\boldsymbol{\chi}) \approx t_\tau(\bar{F}_\tau(\boldsymbol{\chi}))$. The transformation functions $t : \mathbb{R} \to \mathbb{R}$ are one-dimensional mappings and it seems fully sufficient to use for instance piece-wise linear functions (PLiFs) $p_{\boldsymbol{\mu},\boldsymbol{\theta}}(\lambda) := \langle \boldsymbol{\varphi}_{\boldsymbol{\mu}}(\lambda), \boldsymbol{\theta} \rangle$ with fixed abscissa boundaries $\boldsymbol{\mu}$ and $\boldsymbol{\theta}$-parametrized ordinate values ($\boldsymbol{\varphi}_{\boldsymbol{\mu}}(\lambda)$ can be appropriately defined). We may define the mapping $\Phi(\boldsymbol{x}, \boldsymbol{y})$ for our case as

$$
\Phi(\boldsymbol{x},\boldsymbol{y}) = \left( \begin{array}{c} \left( \sum_{j=1}^{s(\boldsymbol{y})} [\![ v_j = \sigma ]\!] [\![ v_{j+1} = \tau ]\!] \, \boldsymbol{\varphi}_{\mu_{\sigma,\tau}}(\bar{F}_{\sigma,\tau}(\chi_{\pi_j..\pi_{j+1}})) \right)_{\sigma,\tau\in\Upsilon} \\ \left( \sum_{j=1}^{s(\boldsymbol{y})} [\![ v_{j+1} = \tau ]\!] \, \boldsymbol{\varphi}_{\mu_\tau}(\bar{F}_\tau(\chi_{\pi_{j+1}\pm\omega})) \right)_{\tau\in\Upsilon} \end{array} \right), \tag{7}
$$

where we simply replaced the feature with PLiF features based on the outcomes of precomputed predictions. Note that $\Phi(\boldsymbol{x}, \boldsymbol{y})$ has only $(|\Upsilon|^2 + |\Upsilon|)P$ dimensions, where $P$ is the number of support points used in the PLiFs.

If the alphabet $\Upsilon$ is reasonably small then the dimensionality is low enough to solve the optimization problem (2) efficiently in the primal domain. In the next section we will illustrate how to successfully apply a version of the outlined algorithm to a problem where we have several thousands of relatively long labeled sequences.

## 4 Application to Gene Structure Prediction

The problem of gene structure prediction is to segment nucleotide sequences (so-called pre-mRNA sequences generated by transcription; cf. Figure 4) into exons and introns. In a complex biochemical process called splicing the introns are removed from the pre-mRNA sequence to form the mature mRNA sequence that can be translated into protein. The exon-intron and intron-exon boundaries are defined by sequence motifs almost always containing the letters GT and AG (cf. Figure 4), respectively. However, these dimers appear very frequently and one needs sophisticated methods to recognize true splice sites [21, 12, 13].

So far mostly HMM-based methods such as Genscan [5], Snap [8] or ExonHunter [4] have been applied to this problem and also to the more difficult problem of gene finding. In this work we show

that our newly developed method is applicable to this task and achieves very competitive results. We call it *mSplicer*. Figure 2 illustrates the "grammar" that we use for gene structure prediction. We only require four different states (start, exon-end, exon-start and end) and two different segment labels (exon & intron). Biologically it makes sense to distinguish between first, internal, last and single exons, as their typical lengths are quite different. Each of these exon types correspond to one transition in the model. States two and three recognize the two types of splice sites and the transition between these states defines an intron.

For our specific problem we only need signal detectors for segments ending in state two and three. In the next subsection we outline how we obtain $\bar{F}_2$ and $\bar{F}_3$. Additionally we need content sensors for every possible transition. While the "content" of the different exon segments is essentially the same, the length of them can vary quite drastically. We therefore decided to use one content sensor $\bar{F}_I$ for the intron transition $2 \to 3$ and the same content sensor $\bar{F}_E$ for all four exon transitions $1 \to 2, 1 \to 4, 3 \to 2$ and $3 \to 4$. However, in order to capture the different length characteristics, we include

$$\left( \sum_{j=1}^{s(\boldsymbol{y})} [\![v_j = \sigma]\!] [\![v_{j+1} = \tau]\!] \boldsymbol{\varphi}_{\boldsymbol{\gamma}_{\sigma,\tau}} (\pi_{j+1} - \pi_j) \right)_{\sigma,\tau \in \Upsilon} \tag{8}$$

in the feature map (7), which amounts to using PLiFs for the lengths of all transitions. Also, note that we can drop those features in (7) and (8) that correspond to transitions that are not allowed (e.g. $4 \to 1$; cf. Figure 2).[5]

We have obtained data for training, validation and testing from public sequence databases (see [13] for details).For the considered genome of *C. elegans* we have split the data into four different sets: Set 1 is used for training the splice site signal detectors and the two content sensors; Set 2 is used for model selection of the latter signal detectors and content sensors and for training the HSM SVM; Set 3 is used for model selection of the HSM SVM; and Set 4 is used for the final evaluation. These are large scale datasets, with which current Hidden-Markov-SVMs are unable to deal with: The *C. elegans* training set used for label-sequence learning contains 1,536 sequences with an average length of $\approx 2,300$ base pairs and about 9 segments per sequence, and the splice site signal detectors where trained on more than a million examples. In principle it is possible to join sets 1 & 2, however, then the predictions of $\bar{F}_{\sigma,\tau}$ and $\bar{F}_\tau$ on the sequences used for the HSM SVM are skewed in the margin area (since the examples are pushed away from the decision boundary on the training set). We therefore keep the two sets separated.

## 4.1 Learning the Splice Site Signal Detectors

From the training sequences (Set 1) we extracted sequences of confirmed splice sites (intron start and end). For intron start sites we used a window of $[-80, +60]$ around the site. For intron end sites we used $[-60, +80]$. From the training sequences we also extracted non-splice sites, which are within an exon or intron of the sequence and have `AG` or `GT` consensus. We train an SVM [19] with soft-margin using the WD kernel [12]: $\mathrm{k}(\mathbf{x}, \mathbf{x}') = \sum_{j=1}^{d} \beta_j \sum_{i=1}^{l-j} [\![(x_{[i,i+j]} = x'_{[i,i+j]})]\!]$, where $l = 140$ is the length of the sequence and $x_{[a,b]}$ denotes the sub-string of $x$ from position $a$ to (excluding) $b$ and $\beta_j := d - j + 1$. We used a normalization of the kernel $\tilde{k}(\boldsymbol{x}, \boldsymbol{x}') = \frac{k(\boldsymbol{x},\boldsymbol{x}')}{\sqrt{k(\boldsymbol{x},\boldsymbol{x})k(\boldsymbol{x}',\boldsymbol{x}')}}$. This leads to the two discriminative functions $\bar{F}_2$ and $\bar{F}_3$. All model parameters (including the window size) have been tuned on the validation set (Set 2). SVM training for *C. elegans* resulted in 79,000 and 61,233 support vectors for detecting intron start and end sites, respectively.

[5]We also excluded these transitions during the Viterbi-like algorithm.

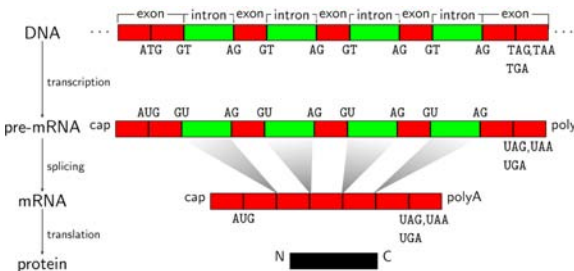

Figure 1: The major steps in protein synthesis [10]. A transcript of a gene starts with an exon and may then be interrupted by an *intron*, followed by another exon, intron and so on until it ends in an exon. In this work we learn the unknown formal mapping from the pre-mRNA to the mRNA.

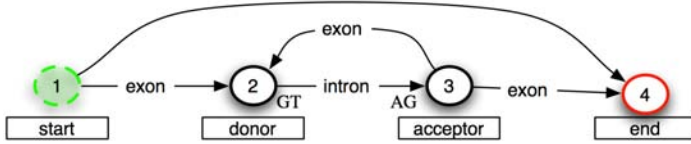

Figure 2: An elementary state model for unspliced mRNA: The start is either directly followed by the end or by an arbitrary number of donor-acceptor splice site pairs.

## 4.2 Learning the Exon and Intron Content Sensors

To obtain the exon content sensor we derived a set of exons from the training set. As negative examples we used sub-sequences of intronic sequences sampled such that both sets of strings have roughly the same length distribution. We trained SVMs using a variant of the *Spectrum kernel* [21] of degree $d = 6$, where we count 6-mers appearing at least once in both sequences. We applied the same normalization as in Sec. 4.1 and proceeded analogously for the intron content sensor. The model parameters have been obtained by tuning them on the validation set.

Note that the resulting content sensors $\bar{F}_I$ and $\bar{F}_E$ need to be evaluated several times during the Viterbi-like algorithm (cf. (6)): One needs to extend segments ending at the same position $i$ to several different starting points. By re-using the shorter segment's outputs this computation can be made drastically faster.

## 4.3 Combination

For datasets 2-4 we can precompute all candidate splice sites using the classifiers $\bar{F}_2$ and $\bar{F}_3$. We decided to use PLiFs with $P = 30$ support points and chose the boundaries for $\bar{F}_2$, $\bar{F}_3$, $\bar{F}_E$, and $\bar{F}_I$ uniformly between $-5$ and $5$ (typical range of outputs of our SVMs). For the PLiFs concerned with length of segments we chose appropriate boundaries in the range $30 - 1000$. With all these definitions the feature map as in (7) and (8) is fully defined. The model has nine PLiFs as parameters, with a total of 270 parameters.

Finally, we have modified the regularizer for our particular case, which favors smooth PLiFs:

$$\mathbf{P}(\boldsymbol{w}) := \sum_{\sigma,\tau\in\Upsilon} |w_P^{\sigma,\tau} - w_1^{\sigma,\tau}| + \sum_{\tau\in\Upsilon} |w_P^{\tau} - w_1^{\tau}| + \sum_{\tau\in\Upsilon} \sum_{i=1}^{P-1} |w_i^{\tau,l} - w_{i+1}^{\tau,l}|,$$

where $\boldsymbol{w} = \left((\boldsymbol{w}^{\sigma,\tau})_{\sigma,\tau\in\Upsilon}; (\boldsymbol{w}^{\tau})_{\tau\in\Upsilon}; (\boldsymbol{w}^{\tau,l})_{\tau\in\Upsilon}\right)$ and we constrain the PLiFs for the signal and content sensors to be monotonically increasing.[6]

Having defined the feature map and the regularizer, we can now apply the HSM SVM algorithm outlined in Sections 2.4 and 3. Since the feature space is rather low dimensional (270 dimensions), we can solve the optimization problem in the primal domain even with several thousands of examples employing a standard optimizer (we used ILOG CPLEX and column generation) within a reasonable time.[7]

## 4.4 Results

To estimate the out-of-sample accuracy, we apply our method to the independent test dataset 4. For *C. elegans* we can compare it to *ExonHunter*[8] on 1177 test sequences. We greatly outperform the *ExonHunter* method: our method obtains almost 1/3 of the test error of *ExonHunter* (cf. Table 1). Simplifying the problem by only considering sequences between the start and stop codons allows us to also include *SNAP* in the comparison on the dataset 4', a slightly modified version of dataset 4 with 1138 sequences.[9] The results are shown in Table 1. On dataset 4' the best competing method achieves an error rate of 9.8% which is more than twice the error rate of our method.

## 5 Conclusion

We have extended the framework of Hidden Markov SVMs to Hidden Semi-Markov SVMs and suggested an very efficient two-stage learning algorithm to train an approximation to Hidden Semi-Markov SVMs. Moreover, we have successfully applied our method on large scale gene structure

| Method | C. elegans Dataset 4 | | | | |
|---|---|---|---|---|---|
| | error rate | exon Sn | exon Sp | exon nt Sn | exon nt Sp |
| *Our Method* | **13.1%** | **96.7%** | **96.8%** | **98.9%** | 97.2% |
| *ExonHunter* | 36.8% | 89.1% | 88.4% | 98.2% | **97.4%** |
| | C. elegans Dataset 4' | | | | |
| *Our Method** | **4.8%** | **98.9%** | **99.2%** | 99.2% | **99.9%** |
| *ExonHunter** | 9.8% | 97.9% | 96.6% | **99.4%** | 98.1% |
| *SNAP** | 17.4% | 95.0% | 93.3% | **99.0%** | 98.9% |

Table 1: Shown are the rates of predicting a wrong gene structure, sensitivity (Sn) and specificity (Sp) on exon and nucleotide levels (see e.g. [8]) for our method, *ExonHunter* and *SNAP*. The methods exploiting additional biological knowledge have an advantage and are marked with *.

prediction appearing in computational biology, where our method obtains less than a half of the error rate of the best competing HMM-based method. Our predictions are available at Wormbase: `http://www.wormbase.org`. Additional data and results are available at the project's website `http://www.fml.mpg.de/raetsch/projects/msplicer`.

**Acknowledgments**   We thank K.-R. Müller, B. Schölkopf, E. Georgii, A. Zien, G. Schweikert and G. Zeller for inspiring discussions. The latter three we also thank for proofreading the manuscript. Moreover, we thank D. Surendran for naming the piece-wise linear functions *PLiF* and optimizing the Viterbi-implementation.

## Footnotes

[1]Note that the number of possible labelings grows exponentially in the length of the sequence.

[2] We define $v_{l+1} := \emptyset$ to keep the notation simple.

[3] Note that one can extend the outlined decoding algorithm to produce not only the best path, but the $K$ best paths. The 2$^{\text{nd}}$ best path may be required to compute the structure in (3). The idea is to duplicate tables $K$ times as follows:

[6]This implements our intuition that large SVM scores should lead to larger scores for a labeling.

[7]It takes less than one hour to solve the HSM SVM problem with about 1,500 sequences on a single CPU. Training the content and signal detectors on several hundred thousand examples takes around 5 hours in total.

[8]The method was trained by their authors on the same training data.

[9]In this setup additional biological information about the so-called "open reading frame" is used: As there was only a version of *SNAP* available that uses this information, we incorporated this extra knowledge also in our model (marked *) and also used another version of *Exonhunter* that also exploits that information in order to allow a fair comparison.

# References

[1] Y. Altun, T. Hofmann, and A. Smola. Gaussian process classification for segmenting and annotating sequences. In *Proc. ICML 2004*, 2004.

[2] Y. Altun, D. McAllester, and M. Belkin. Maximum margin semi-supervised learning for structured variables. In *Proc. NIPS 2005*, 2006.

[3] Y. Altun, I. Tsochantaridis, and T. Hofmann. Hidden Markov support vector machines. In T. Fawcett, editor, *Proc. 20th Int. Conf. Mach. Learn.*, pages 3–10, 2003.

[4] B. Brejova, D.G. Brown, M. Li, and T. Vinar. ExonHunter: a comprehensive approach to gene finding. *Bioinformatics*, 21(Suppl 1):i57–i65, 2005.

[5] C. Burge and S. Karlin. Prediction of complete gene structures in human genomic DNA. *Journal of Molecular Biology*, 268:78–94, 1997.

[6] X. Ge. *Segmental Semi-Markov Models and Applications to Sequence Analysis*. PhD thesis, University of California, Irvine, 2002.

[7] J. Janssen and N. Limnios. *Semi-Markov Models and Applications*. Kluwer Academic, 1999.

[8] I. Korf. Gene finding in novel genomes. *BMC Bioinformatics*, 5(59), 2004.

[9] D. Kulp, D. Haussler, M.G. Reese, and F.H. Eeckman. A generalized hidden markov model for the recognition of human genes in DNA. *ISMB 1996*, pages 134–141, 1996.

[10] B. Lewin. *Genes VII*. Oxford University Press, New York, 2000.

[11] L.R. Rabiner. A tutorial on hidden markov models and selected applications in speech recognition. *Proceedings of the IEEE*, 77(2):257–285, February 1989.

[12] G. Rätsch and S. Sonnenburg. Accurate splice site prediction for Caenorhabditis elegans. In B. Schölkopf, K. Tsuda, and J.-P. Vert, editors, *Kernel Methods in Computational Biology*. MIT Press, 2004.

[13] G. Rätsch, S. Sonnenburg, J. Srinivasan, H. Witte, K.-R. Müller, R. Sommer, and B. Schölkopf. Improving the C. elegans genome annotation using machine learning. *PLoS Computational Biology*, 2007. In press.

[14] S. Sarawagi and W.W. Cohen. Semi-markov conditional random fields for information extraction. In *Proc. NIPS 2004*, 2005.

[15] G.D. Stormo and D. Haussler. Optimally parsing a sequence into different classes based on multiple types of information. In *Proc. ISMB 1994*, pages 369–375, Menlo Park, CA, 1994. AAAI/MIT Press.

[16] R. Sutton, D. Precup, and S. Singh. Between mdps and semi-mdps: A framework for temporal abstraction in reinforcement learrning. *Artificial Intelligence*, 112:181–211, 1999.

[17] B. Taskar, C. Guestrin, and D. Koller. Max-margin markov networks. In *Proc. NIPS 2003*, 16, 2004.

[18] I. Tsochantaridis, T. Hofmann, T. Joachims, and Y. Altun. Large margin methods for structured output spaces. *Journal for Machine Learning Research*, 6, September 2005.

[19] V.N. Vapnik. *The nature of statistical learning theory*. Springer Verlag, New York, 1995.

[20] A. J. Viterbi. Error bounds for convolutional codes and an asymptotically optimal decoding algorithm. *IEEE Trans. Informat. Theory*, IT-13:260–269, Apr 1967.

[21] X.H. Zhang, K.A. Heller, I. Hefter, C.S. Leslie, and L.A. Chasin. Sequence information for the splicing of human pre-mRNA identified by SVM classification. *Genome Res*, 13(12):2637–50, 2003.